# Learning to Find Pre-Images

**Gökhan H. Bakır, Jason Weston and Bernhard Schölkopf**
Max Planck Institute for Biological Cybernetics
Spemannstraße 38, 72076 Tübingen, Germany
{gb,weston,bs}@tuebingen.mpg.de

## Abstract

We consider the problem of reconstructing patterns from a feature map. Learning algorithms using kernels to operate in a reproducing kernel Hilbert space (RKHS) express their solutions in terms of input points mapped into the RKHS. We introduce a technique based on kernel principal component analysis and regression to reconstruct corresponding patterns in the input space (aka pre-images) and review its performance in several applications requiring the construction of pre-images. The introduced technique avoids difficult and/or unstable numerical optimization, is easy to implement and, unlike previous methods, permits the computation of pre-images in discrete input spaces.

## 1 Introduction

We denote by $\mathcal{H}_k$ the RKHS associated with the kernel $k(\mathbf{x}, \mathbf{y}) = \phi(\mathbf{x})^\top \phi(\mathbf{y})$, where $\phi(\mathbf{x}) : \mathcal{X} \to \mathcal{H}_k$ is a possible nonlinear mapping from input space $\mathcal{X}$ (assumed to be a nonempty set) to the possible infinite dimensional space $\mathcal{H}_k$. The pre-image problem is defined as follows: given a point $\mathbf{\Psi}$ in $\mathcal{H}_k$, find a corresponding pattern $\mathbf{x} \in \mathcal{X}$ such that $\mathbf{\Psi} = \phi(\mathbf{x})$. Since $\mathcal{H}_k$ is usually a far larger space than $\mathcal{X}$, this is often not possible (see Fig. **??**). In these cases, the (approximate) pre-image $\mathbf{z}$ is chosen such that the squared distance of $\Psi$ and $\phi(\mathbf{z})$ is minimized,

$$\mathbf{z} = \arg\min_{\mathbf{z}} \|\mathbf{\Psi} - \phi(\mathbf{z})\|^2. \tag{1}$$

This has a significant range of applications in kernel methods: for *reduced set methods* [1], for denoising and compression using kernel principal components analysis (kPCA), and for kernel dependency estimation (KDE), where one finds a mapping between paired sets of objects. The techniques used so far to solve this nonlinear optimization problem often employ gradient descent [1] or nonlinear iteration methods [2]. Unfortunately, this suffers from (i) being a difficult nonlinear optimization problem with local minima requiring restarts and other numerical issues, (ii) being computationally inefficient, given that the problem is solved individually for each testing example, (iii) not being the optimal approach (e.g., we may be interested in minimizing a classification error rather then a distance in feature space); and (iv) not being applicable for pre-images which are objects with discrete variables.

In this paper we propose a method which can resolve all four difficulties: the simple idea is to estimate the function (1) by learning the map $\Psi \to \mathbf{z}$ from examples $(\phi(\mathbf{z}), \mathbf{z})$. Depending on the learning technique used this can mean, after training, each use of the function

(each pre-image found) can be computed very efficiently, and there are no longer issues with complex optimization code. Note that this problem is unusual in that it is possible to produce an infinite amount of training data ( and thus expect to get good performance) by generating points in $\mathcal{H}_k$ and labeling them using (1). However, often we have knowledge about the distribution over the pre-images, e.g., when denoising digits with kPCA, one expects as a pre-image something that looks like a digit, and an estimate of this distribution is actually given by the original data. Taking this distribution into account, it is conceivable that a learning method could outperform the naive method, that of equation (1), by producing pre-images that are subjectively preferable to the minimizers of (1). Finally, learning to find pre-images can also be applied to objects with discrete variables, such as for string outputs as in part-of-speech tagging or protein secondary structure prediction.

The remainder of the paper is organized as follows: in Section 2 we review kernel methods requiring the use of pre-images: kPCA and KDE. Then, in Section 3 we describe our approach for learning pre-images. In Section 4 we verify our method experimentally in the above applications, and in Section 5 we conclude with a discussion.

## 2 Methods Requiring Pre-Images

### 2.1 Kernel PCA Denoising and Compression

Given data points $\{\mathbf{x}_i\}_{i=1}^m \in \mathcal{X}$, kPCA constructs an orthogonal set of feature extractors in the RKHS. The constructed orthogonal system $P = \{\mathbf{v}_1, \ldots, \mathbf{v}_r\}$ lies in the span of the data points, i.e., $P = \left(\sum_{i=1}^m \alpha_i^1 \phi(\mathbf{x}_i), \ldots, \sum_{i=1}^m \alpha_i^r \phi(\mathbf{x}_i)\right)$. It is obtained by solving the eigenvalue problem $m\lambda\alpha^i = \mathbf{K}\alpha^i$ for $1 \leq i \leq r$ where $K_{ij} = k(\mathbf{x}_i, \mathbf{x}_j)$ is the kernel matrix and $r \leq m$ is the number of nonzero eigenvalues.[1] Once built, the orthogonal system $P$ can be used for nonlinear feature extraction. Let $\mathbf{x}$ denote a test point, then the nonlinear principal components can be extracted via $P\phi(\mathbf{x}) = \left(\sum_{i=1}^m \alpha_i^1 k(\mathbf{x}_i, \mathbf{x}), \ldots, \sum_{i=1}^m \alpha_i^r k(\mathbf{x}_i, \mathbf{x})\right)$ where $k(\mathbf{x}_i, \mathbf{x})$ is substituted for $\phi(\mathbf{x}_i)^\top \phi(\mathbf{x})$. See ([3],[4] chapter 14) for details.

Beside serving as a feature extractor, kPCA has been proposed as a denoising and compression procedure, both of which require the calculation of input patterns $\mathbf{x}$ from feature space points $P\phi(\mathbf{x})$.

**Denoising.** Denoising is a technique used to reconstruct patterns corrupted by noise. Given data points $\{\mathbf{x}_i\}_{i=1}^m$ and the orthogonal system $P = (\mathbf{v}_1, \ldots, \mathbf{v}_a, \ldots, \mathbf{v}_r)$ obtained by kPCA. Assume that the orthogonal system is sorted by decreasing variance, we write $\phi(\mathbf{x}) = P\phi(\mathbf{x}) = P_a\phi(\mathbf{x}) + P_a^\perp\phi(\mathbf{x})$, where $P_a$ denotes the projection on the span of $(\mathbf{v}_1, \ldots, \mathbf{v}_a)$. The hope is that $P_a\phi(\mathbf{x})$ retains the main structure of $\mathbf{x}$, while $P_a^\perp\phi(\mathbf{x})$ contains noise. If this is the case, then we should be able to construct a denoised input pattern as the pre-image of $P_a\phi(\mathbf{x})$. This *denoised* pattern $\mathbf{z}$ can be obtained as solution to the problem

$$\mathbf{z} = \arg\min_{\mathbf{z}} \|P_a\phi(\mathbf{x}) - \phi(\mathbf{z})\|^2. \tag{2}$$

For an application of kPCA denoising see [2].

**Compression.** Consider a sender receiver-scenario, where the sender S wants to transmit information to the receiver R. If S and R have the same projection matrix $P$ serving as a vocabulary, then S could use $P_a$ to encode $\mathbf{x}$ and send $P_a\phi(\mathbf{x}) \in \mathbb{R}^a$ instead of $\mathbf{x} \in \mathbb{R}^n$. This corresponds to a lossy compression, and is useful if $a \ll n$. R would obtain the

corresponding pattern $\mathbf{x}$ by minimizing (2) again. Therefore kPCA would serve as encoder and the pre-image technique as decoder.

## 2.2 Kernel Dependency Estimation

Kernel Dependency Estimation (KDE) is a novel algorithm [5] which is able to learn general mappings between an input set $\mathcal{X}$ and output set $\mathcal{Y}$, give definitions of kernels $k$ and $l$ (with feature maps $\Phi_k$ and $\Phi_l$) which serve as similarity measures on $\mathcal{X}$ and $\mathcal{Y}$, respectively. To learn the mapping from data $\{\mathbf{x}_i, \mathbf{y}_i\}_{i=1}^m \in \mathcal{X} \times \mathcal{Y}$, KDE performs two steps.

**1) Decomposition of outputs.** First a kPCA is performed in $\mathcal{H}_l$ associated with kernel $l$. This results in $r$ principal axes $\mathbf{v}_1, \ldots, \mathbf{v}_r$ in $\mathcal{H}_l$. Obtaining the principal axes, one is able to obtain principal components $(\phi_l(\mathbf{y})^\top \mathbf{v}_1, \ldots, \phi_l(\mathbf{y})^\top \mathbf{v}_r)$ of any object $\mathbf{y}$.

**2) Learning the map.** Next, we learn the map from $\phi_k(\mathbf{x})$ to $(\phi_l(\mathbf{y})^\top \mathbf{v}_1, \ldots, \phi_l(\mathbf{y})^\top \mathbf{v}_r)$. To this end, for each principal axis $\mathbf{v}_j$ we solve the problem

$$\arg\min_{\beta^j} \sum_{i=1}^m (\phi_l(y_i)^\top \mathbf{v}_j - g(\mathbf{x}_i, \beta^j))^2 + \gamma \|\beta^j\|^2, \tag{3}$$

where $\gamma \|\beta^j\|^2$ acts as a regularization term (with $\gamma > 0$), $g(\mathbf{x}_i, \beta^j) = \sum_{s=1}^m \beta_s^j k(\mathbf{x}_s, \mathbf{x}_i)$, and $\beta \in \mathbb{R}^{m \times r}$. Let $\mathbf{P} \in \mathbb{R}^{m \times r}$ with $P_{ij} = \phi_l(\mathbf{y}_i)^\top \mathbf{v}_j, j = 1 \ldots r$ and $\mathbf{K} \in \mathbb{R}^{m \times m}$ the kernel matrix with entries $K_{st} = k(\mathbf{x}_s, \mathbf{x}_t)$, with $s, t = 1 \ldots m$. Problem (3) can then be minimized, for example via kernel ridge regression, yielding

$$\beta = (\mathbf{K}^\top \mathbf{K} + \gamma \mathbf{I})^{-1} \mathbf{K} \mathbf{P}. \tag{4}$$

**3) Testing Phase.** Using the learned map from input patterns to principal components, predicting an output $\mathbf{y}'$ for a new pattern $\mathbf{x}'$ requires solving the pre-image problem

$$\mathbf{y}' = \arg\min_y \|(\phi_l(\mathbf{y})^\top \mathbf{v}_1, \ldots, \phi_l(\mathbf{y})^\top \mathbf{v}_r) - (k(\mathbf{x}_1, \mathbf{x}'), \ldots, k(\mathbf{x}_m, \mathbf{x}'))\beta\|^2. \tag{5}$$

Thus $\mathbf{y}'$ is the approximate pre-image of the estimated point $\phi(\mathbf{y}')$ in $\mathcal{H}_l$.

## 3 Learning Pre-Images

We shall now argue that by mainly being concerned with (1), the methods that have been used for this task in the past disregard an important piece of information. Let us summarize the state of the art (for details, see [4]).

**Exact pre-images.** One can show that if an exact pre-image exists, and if the kernel can be written as $k(\mathbf{x}, \mathbf{x}') = f_k((\mathbf{x}^\top \mathbf{x}'))$ with an invertible function $f_k$ (e.g., $k(\mathbf{x}, \mathbf{x}') = (\mathbf{x}^\top \mathbf{x}')^d$ with odd $d$), then one can compute the pre-image analytically as $\mathbf{z} = \sum_{i=1}^N f_k^{-1}\left(\sum_{j=1}^m \alpha_j k(\mathbf{x}_j, \mathbf{e}_i)\right) \mathbf{e}_i$, where $\{\mathbf{e}_1, \ldots, \mathbf{e}_N\}$ is any orthonormal basis of input space. However, if one tries to apply this method in practice, it usually works less well than the approximate pre-image methods described below. This is due to the fact that it usually is not the case that exact pre-images exist.

**General approximation methods.** These methods are based on the minimization of (1). Whilst there are certain cases where the minimizer of (1) can be found by solving an eigenvalue problem (for $k(\mathbf{x}, \mathbf{x}') = (\mathbf{x}^\top \mathbf{x}')^2$), people in general resort to methods of nonlinear optimization. For instance, if the kernel is differentiable, one can multiply out (1) to express it in terms of the kernel, and then perform gradient descent [1]. The drawback of these methods is that the optimization procedure is expensive and will in general only find a local optimum. Alternatively one can select the k best input points from some training set and use them in combination to minimize the distance (1), see [6] for details.

**Iteration schemes for particular kernels.** For particular types of kernels, such as radial basis functions, one can devise fixed point iteration schemes which allow faster minimization of (1). Again, there is no guarantee that this leads to a global optimum.

One aspect shared by all these methods is that they do not explicitly make use of the fact that we have *labeled examples* of the unknown pre-image map: specifically, if we consider any point in $\mathbf{x} \in \mathcal{X}$, we know that the pre-image of $\Phi(\mathbf{x})$ is simply $\mathbf{x}$.[2] Below, we describe a method which makes heavy use of this information. Specifically, we use kernel regression to estimate the pre-image map from data. As a data set, we consider the training data $\{\mathbf{x}_i\}_{i=1}^m$ that we are given in our original learning problem (kPCA, KDE, etc.).

## 3.1 Estimation of the Pre-Image Map

We seek to estimate a function $\Gamma : \mathcal{H}_k \to \mathcal{X}$ with the property that, at least approximately, $\Gamma(\Phi(\mathbf{x}_i)) = \mathbf{x}_i$ for $i = 1, \ldots, m$. If we were to use regression using the kernel $k$ corresponding to $\mathcal{H}_k$, then we would simply look for weight vectors $\mathbf{w}_j \in \mathcal{H}_k$, $j = 1, \ldots, \dim \mathcal{X}$ such that $\Gamma_j(\Psi) = \mathbf{w}_j^\top \Psi$, and use the kernel trick to evaluate $\Gamma$. However, in general we may want to use a kernel $\kappa$ which is different from $k$, and thus we cannot perform our computations implicitly by the use of a kernel. This looks like a problem, but there is a way to handle it. It is based on the well-known observation that although the data in $\mathcal{H}_k$ may live in an infinite-dimensional space, any finite data set spans a subspace of finite dimension. A convenient way of working in that subspace is to choose a basis and to work in coordinates, e.g., using a kPCA basis. Let $P_n \Psi = \sum_{i=1}^n (\Psi^\top \mathbf{v}_i) \mathbf{v}_i$ denote the projection that maps a point into its coordinates in the PCA basis $\mathbf{v}_1, \ldots, \mathbf{v}_n$, i.e., into the subspace where the training set has nonzero variance. We then learn the pre-image map $\Gamma_j : \mathbb{R}^n \to \mathcal{X}$ by solving the learning problem

$$\Gamma_j = \operatorname*{arg\,min}_{\Gamma_j} \sum_{i=1}^m l\left(\mathbf{x}_i, \Gamma(P_n \phi(\mathbf{x}_i))\right) + \lambda \Omega(\Gamma). \tag{6}$$

Here, $\Omega$ is a regularizer, and $\lambda \geq 0$. If $\mathcal{X}$ is the vector space $\mathbb{R}^N$, we can consider the problem (6) as a standard regression problem for the $m$ training points $\mathbf{x}_i$ and use kernel ridge regression with a kernel $\kappa$. This yields a pre-image mapping $\Gamma_j(P_n \phi(\mathbf{x})) = \sum_{r=1}^m \beta_r^j \kappa(P_n \phi(\mathbf{x}), P_n \phi(\mathbf{x}_r))$, $j = 1, \ldots, N$, which can be solved like (3).

Note that the general learning setup of (6) allows to use of any suitable loss function, incorporating invariances and a-priori knowledge. For example, if the pre-images are (natural) images, a psychophysically motivated loss function could be used, which would allow the algorithm to ignore differences that cannot be perceived.

## 3.2 Pre-Images for Complex Objects

In methods such as KDE one is interested in finding pre-images for general sets of objects, e.g. one may wish to find a string which is the pre-image of a representation using a string kernel [7, 8]. Using gradient descent techniques this is not possible as the objects have discrete variables (elements of the string). However, using function estimation techniques, as long as it is possible to learn to find pre-images even for such objects, the problem can be approached by decomposition into several learning subtasks. This should be possible whenever there is structure in the object one is trying to predict. In the case of strings one can predict each character of the string independently given the estimate $\phi_l(\mathbf{y}')$. This is made particularly tractable in fixed-length string prediction problems such as for part-of-speech tagging or protein secondary structure prediction because the length is known (it is the same length as the input). Otherwise the task is more difficult but still one could also

predict the length of the output string before predicting each element of it. As an example, we now describe in depth a method for finding pre-images for known-length strings.

The task is to predict a string $\mathbf{y}$ given a string $\mathbf{x}$ and a set of paired examples $(\mathbf{x}_i, \mathbf{y}_i) \in \cup_{p=1}^{\infty} (\Sigma_x)^p \times \cup_{p=1}^{\infty} (\Sigma_y)^p$. Note that $|\mathbf{x}_i| = |\mathbf{y}_i|$ for all $i$, i.e., the length of any paired input and output strings are the same. This is the setting of part-of-speech tagging, where $\Sigma_x$ are words and $\Sigma_y$ are parts of speech, and also secondary structure prediction, where $\Sigma_x$ are amino acids of a protein sequence and $\Sigma_y$ are classes of structure that the sequence folds into, e.g. helix, sheet or coil.

It is possible to use KDE (Section 2.2) to solve this task directly. One has to define an appropriate similarity function for both sets of objects using a kernel function, giving two implicit maps $\phi_k(\mathbf{x})$ and $\phi_l(\mathbf{y})$ using string kernels. KDE then learns a map between the two feature spaces, and for a new test string $\mathbf{x}$ one must find the pre-image of the estimate $\phi_l(\mathbf{y}')$ as in equation (5). One can find this pre-image by predicting each character of the string independently given the estimate $\phi_l(\mathbf{y}')$ as it has known length given the input $\mathbf{x}$.

One can thus learn a function $b_p = f(\phi_l(y'), \alpha_p)$ where $b_p$ is the $p^{th}$ element of the output and $\alpha_p = (a_{(p-n/2)} a_{(p-n/2+1)} \ldots a_{(p+n/2)})$ is a window of length $n + 1$ with center at position $p$ in the input string. One computes the entire output string with $\beta = (f(\phi_l(y'), \alpha_1) \ldots f(\phi_l(y'), \alpha_{|\mathbf{x}|}))$; window elements outside of the string can be encoded with a special terminal character. The function $f$ can be trained with any multi-class classification algorithm to predict one of the elements of the alphabet, the approach can thus be seen as a generalization of the traditional approach which is learning a function $f$ given only a window on the input (the second parameter). Our approach first estimates the output using global information from the input and with respect to the loss function of interest on the outputs—it only decodes this global prediction in the final step. Note that problems such as secondary structure prediction often have loss functions dependent on the complete outputs, not individual elements of the output string [9].

## 4 Experiments

In the following we demonstrate the pre-image learning technique on the applications we have introduced.

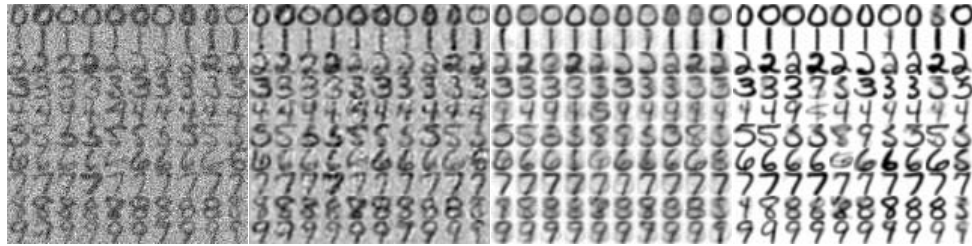

Gaussian noise      PCA      kPCA+grad.desc.      kPCA+learn-pre.

Figure 1: Denoising USPS digits: linear PCA fails on this task, learning to find pre-images for kPCA performs at least as well as finding pre-images by gradient descent.

**KPCA Denoising.** We performed a similar experiment to the one in [2] for demonstration purposes: we denoised USPS digits using linear and kPCA. We added Gaussian noise with variance 0.5 and selected 100 randomly chosen non-noisy digits for training and a further 100 noisy digits for testing, 10 from each class. As in [2] we chose a nonlinear map via a Gaussian kernel with $\sigma = 8$. We selected 80 principal components for kPCA. We found pre-images using the Matlab function fminsearch, and compared this to our pre-

image-learning method (RBF kernel $K(x, x') = \exp(-||x - x'||^2/2\sigma^2)$ with $\sigma = 1$, and regularization parameter $\lambda = 1$). Figure 1 shows the results: our approach appears to perform better than the gradient descent approach. As in [2], linear PCA visually fails for this problem: we show its best results, using 32 components. Note the mean squared error performance of the algorithms is not precisely in accordance with the loss of interest to the user. This can be seen as PCA has an MSE ($13.8\pm0.4$) versus gradient descent ($31.6\pm1.7$) and learnt pre-images ($29.2\pm1.8$). PCA has the lowest MSE but as can be seen in Figure 1 it doesn't give satisfactorys visual results in terms of denoising.

Note that some of the digits shown are actually denoised incorrectly as the wrong class. This is of course possible as choosing the correct digit is a problem which is harder than a standard digit classification problem because the images are noisy. Moreover, kPCA is not a classifier per se and could not be expected to classify digits as well as Support Vector Machines. In this experiment, we also took a rather small number of training examples, because otherwise the fminsearch code for the gradient descent was very slow, and this allowed us to compare our results more easily.

**KPCA Compression.**   For the compression experiment we use a video sequence consisting of 1000 graylevel images, where every frame has a $100 \times 100$ pixel resolution. The video sequence shows a famous science fiction figure turning his head 180 degrees. For training we used every 20th frame resulting in a video sequence of 50 frames with 3.6 degree orientation difference per image. The motivation is to store only these 50 frames and to reconstruct all frames in between.

We applied a kPCA to all 50 frames with a Gaussian kernel with kernel parameter $\sigma_1$. The 50 feature vectors $\mathbf{v}_1, \ldots, \mathbf{v}_{50} \in \mathbb{R}^{50}$ are used then to learn the interpolation between the timeline of the 50 principal components $v_{ij}$ where i is the time index, j the principal component number j and $1 \le i, j \le 50$. A kernel ridge regression with Gaussian kernel and kernel parameter $\sigma_2$ and ridge $r_1$ was used for this task. Finally the pre-image map $\Gamma$ was learned from projections onto $\mathbf{v}_i$ to frames using kernel ridge regression with kernel parameter $\sigma_3$ and ridge $r_2$. All parameters $\sigma_1, \sigma_2, \sigma_3, r_1, r_2$ were selected in a loop such that new synthesized frames looked subjectively best. This led to the values $\sigma_1 = 2.5, \sigma_2 = 1, \sigma_3 = 0.15$ and for the ridge parameters $r_1 = 10^{-13}, r_2 = 10^{-7}$. Figure 2 shows the original and reconstructed video sequence.

Note that the pre-image mechanism could possibly be adapted to take into account invariances and a-priori knowledge like geometries of standard heads to reduce blending effects, making it more powerful than gradient descent or plain linear interpolation of frames. For an application of classical pre-image methods to face modelling, see [10].

**String Prediction with Kernel Dependency Estimation.**   In the following we expose a simple string mapping problem to show the potential of the approach outlined in Section 3.2. We construct an artificial problem with $|\Sigma_x| = 3$ and $|\Sigma_y| = 2$. Output strings are generated by the following algorithm: start in a random state (1 or 2) corresponding to one of the output symbols. The next symbol is either the same or, with probability $\frac{1}{5}$, the state switches (this tends to make neighboring characters the same symbol). The length of the string is randomly chosen between 10 and 20 symbols. Each input string is generated with equal probability from one of two models, starting randomly in state a, b or c and using the following transition matrices, depending on the current output state:

<table>
<tr><td colspan="7" align="center">**Model 1**</td><td colspan="7" align="center">**Model 2**</td></tr>
<tr><td colspan="4">Output 1</td><td colspan="3">Output 2</td><td colspan="4">Output 1</td><td colspan="3">Output 2</td></tr>
<tr><td></td><td>a</td><td>b</td><td>c</td><td>a</td><td>b</td><td>c</td><td></td><td>a</td><td>b</td><td>c</td><td>a</td><td>b</td><td>c</td></tr>
<tr><td>a</td><td>0</td><td>0</td><td>1</td><td>1/2</td><td>1/2</td><td>0</td><td>a</td><td>1/2</td><td>1/2</td><td>0</td><td>1</td><td>0</td><td>0</td></tr>
<tr><td>b</td><td>0</td><td>0</td><td>1</td><td>1/2</td><td>1/2</td><td>0</td><td>b</td><td>1/2</td><td>1/2</td><td>0</td><td>0</td><td>1</td><td>0</td></tr>
<tr><td>c</td><td>1</td><td>0</td><td>0</td><td>0</td><td>1</td><td>0</td><td>c</td><td>0</td><td>1/2</td><td>1/2</td><td>0</td><td>1</td><td>0</td></tr>
</table>

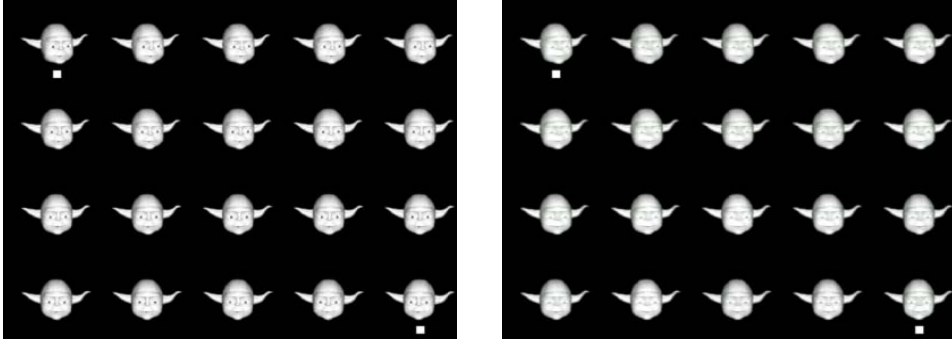

Subsequence of original video sequence.     Subsequence of synthesized video sequence.
First and last frame are used in training set.

Figure 2: Kernel PCA compression used to learn intermediate images. The pre-images are in a $100 \times 100$ dimensional space making gradient-descent based descent impracticable.

As the model of the string can be better predicted from the complete string, a global method could be better in principle than a window-based method. We use a string kernel called the spectrum kernel[11] to define strings for inputs. This method builds a representation which is a frequency count of all possible contiguous subsequences of length $p$. This produces a mapping with features $\phi_k(\mathbf{x}) = \langle \sum_{i=1}^{|\mathbf{x}|-p+1} [(\mathbf{x}_i, \ldots, \mathbf{x}_{(i+p-1)}) = \alpha] : \alpha \in (\Sigma_x)^p \rangle$ where $[\mathbf{x} = \mathbf{y}]$ is 1 if $\mathbf{x} = \mathbf{y}$, and 0 otherwise. To define a feature space for outputs we count the number of contiguous subsequences of length $p$ on the *input* that, if starting in position $q$, have the same element of the alphabet at position $q + (p-1)/2$ in the *output*, for odd values of $p$. That is, $\phi_l(\mathbf{x}, \mathbf{y}) = \langle \sum_{i=1}^{|\mathbf{x}|-p+1} [(\mathbf{x}_i, \ldots, \mathbf{x}_{(i+p-1)}) = \alpha][y_{i+(p-1)/2} = b] : \alpha \in (\Sigma_x)^p, b \in \Sigma_y \rangle$. We can then learn pre-images using a window also of size $p$ as described in Section 3.2, e.g. using $k$-NN as the learner. Note that the output kernel is defined on both the inputs and outputs: such an approach is also used in [12] and called "joint kernels", and in their approach the calculation of pre-images is also required, so they only consider specific kernels for computational reasons. In fact, our approach could also be a benefit if used in their algorithm.

We normalized the input and output kernel matrices such that a matrix $S$ is normalized with $S \leftarrow D^{-1} S D^{-1}$ where $D$ is a diagonal matrix such that $D_i i = \sum_i S_{ii}$. We also used a nonlinear map for KDE, via an RBF kernel, i.e. $K(x, x') = \exp(-d(x, x'))$ where the distance $d$ is induced by the input string kernel defined above, and we set $\lambda = 1$.

We give the results on this toy problem using the classification error (fraction of symbols misclassified) in the table below, with 50 strings using 10-fold cross validation, we compare to $k$-nearest neighbor using a window size of 3, in our method we used $p = 3$ to generate string kernels, and $k$-NN to learn the pre-image, therefore we quote different $k$ for both methods. Results for larger window sizes only made the results worse.

|  | 1-NN | 3-NN | 5-NN | 7-NN | 9-NN |
|---|---|---|---|---|---|
| KDE | 0.182±0.03 | 0.169±0.03 | 0.162±0.03 | 0.164±0.03 | 0.163±0.03 |
| $k$-NN | 0.251±0.03 | 0.243±0.03 | 0.249±0.03 | 0.250±0.03 | 0.248±0.03 |

## 5 Conclusion

We introduced a method to learn the pre-image of a vector in an RKHS. Compared to classical approaches, the new method has the advantage that it is not numerically unstable, it is much faster to evaluate, and better suited for high-dimensional input spaces. It is demon-

strated that it is applicable when the input space is discrete and gradients do not exist. However, as a learning approach, it requires that the patterns used during training reasonably well represent the points for which we subsequently want to compute pre-images. Otherwise, it can fail, an example being a reduced set (see [1]) application, where one needs pre-images of linear combinations of mapped points in $\mathcal{H}$, which can be far away from training points, making generalization of the estimated pre-image map impossible. Indeed, preliminary experiments (not described in this paper) showed that whilst the method can be used to compute reduced sets, it seems inferior to classical methods in that domain.

Finally, the learning of the pre-image can probably be augmented with mechanisms for incorporating a-priori knowledge to enhance performance of pre-image learning, making it more flexible than just a pure optimization approach. Future research directions include the inference of pre-images in structures like graphs and incorporating a-priori knowledge in the pre-image learning stage.

**Acknowledgement.** The authors would like to thank Kwang In Kim for fruitful discussions, and the anonymous reviewers for their comments.

## Footnotes

[1] We assume that the $\phi(\mathbf{x}_i)$ are centered in feature space. This can be achieved by centering the kernel matrix $\mathbf{K}^c = (I - \frac{1}{m}\mathbf{1}\mathbf{1}^\top)K(I - \frac{1}{m}\mathbf{1}\mathbf{1}^\top)$, where $\mathbf{1} \in \mathbb{R}^m$ is the vector with every entry equal 1. Test patterns must be centered with the same center obtained from the training stage.

[2]It may not be the only pre-image, but this does not matter as long as it minimizes the value of (1).

# References

[1] C. J. C. Burges. Simplified support vector decision rules. In L. Saitta, editor, *Proceedings of the 13th International Conference on Machine Learning*, pages 71–77, San Mateo, CA, 1996. Morgan Kaufmann.

[2] S. Mika, B. Schölkopf, A. J. Smola, K.-R. Müller, M. Scholz, and G. Rätsch. Kernel PCA and de-noising in feature spaces. In M. S. Kearns, S. A. Solla, and D. A. Cohn, editors, *Advances in Neural Information Processing Systems 11*, pages 536–542, Cambridge, MA, 1999. MIT Press.

[3] B. Schölkopf, A. J. Smola, and K.-R. Müller. Nonlinear component analysis as a kernel eigenvalue problem. *Neural Computation*, 10:1299–1319, 1998.

[4] B. Schölkopf and A. J. Smola. *Learning with Kernels*. MIT Press, Cambridge, MA, 2002.

[5] Jason Weston, Olivier Chapelle, Andre Elisseeff, Bernhard Schölkopf, and Vladimir Vapnik. Kernel dependency estimation. In S. Becker, S. Thrun, and K. Obermayer, editors, *Advances in Neural Information Processing Systems 15*, Cambridge, MA, 2002. MIT Press.

[6] J.T. Kwok and I.W. Tsang. Finding the pre images in kernel principal component analysis. In *NIPS'2002 Workshop on Kernel Machines*, 2002.

[7] D. Haussler. Convolutional kernels on discrete structures. Technical Report UCSC-CRL-99-10, Computer Science Department, University of California at Santa Cruz, 1999.

[8] H. Lodhi, J. Shawe-Taylor, N. Cristianini, and C. Watkins. Text classification using string kernels. Technical Report 2000-79, NeuroCOLT, 2000. Published in: T. K. Leen, T. G. Dietterich and V. Tresp (eds.), *Advances in Neural Information Processing Systems 13*, MIT Press, 2001, as well as in JMLR **2**:419-444, 2002.

[9] S. Hua and Z. Sun. A novel method of protein secondary structure prediction with high segment overlap measure: Svm approach. *Journal of Molecular Biology*, 308:397–407, 2001.

[10] S. Romdhani, S. Gong, and A. Psarrou. A multiview nonlinear active shape model using kernel PCA. In *Proceedings of BMVC*, pages 483–492, Nottingham, UK, 1999.

[11] C. Leslie, E. Eskin, and W. S. Noble. The spectrum kernel: A string kernel for SVM protein classification. *Proceedings of the Pacific Symposium on Biocomputing*, 2002.

[12] Y. Altun, I. Tsochantaridis, and T. Hofmann. Hidden markov support vector machines. In *20th International Conference on Machine Learning (ICML)*, 2003.
